# Fast, Robust Adaptive Control by Learning only Forward Models

**Andrew W. Moore**
MIT Artificial Intelligence Laboratory
545 Technology Square, Cambridge, MA 02139
awm@ai.mit.edu

## Abstract

A large class of motor control tasks requires that on each cycle the controller is told its current state and must choose an action to achieve a specified, state-dependent, goal behaviour. This paper argues that the optimization of *learning rate*, the number of experimental control decisions before adequate performance is obtained, and *robustness* is of prime importance—if necessary at the expense of computation per control cycle and memory requirement. This is motivated by the observation that a robot which requires two thousand learning steps to achieve adequate performance, or a robot which occasionally gets stuck while learning, will always be undesirable, whereas moderate computational expense can be accommodated by increasingly powerful computer hardware. It is not unreasonable to assume the existence of inexpensive 100 Mflop controllers within a few years and so even processes with control cycles in the low tens of milliseconds will have millions of machine instructions in which to make their decisions. This paper outlines a learning control scheme which aims to make effective use of such computational power.

## 1 MEMORY BASED LEARNING

Memory-based learning is an approach applicable to both classification and function learning in which all experiences presented to the learning box are explicitly remembered. The memory, **Mem**, is a set of input-output pairs, **Mem** = $\{(x_1, y_1), (x_2, y_2), \ldots, (x_k, y_k)\}$. When a prediction is required of the output of a novel input $x_{\text{query}}$, the memory is searched to obtain experiences with inputs close to $x_{\text{query}}$. These local neighbours are used to determine a locally consistent output for the query. Three memory-based techniques, Nearest Neighbour, Kernel Regression, and Local Weighted Regression, are shown in the accompanying figure.

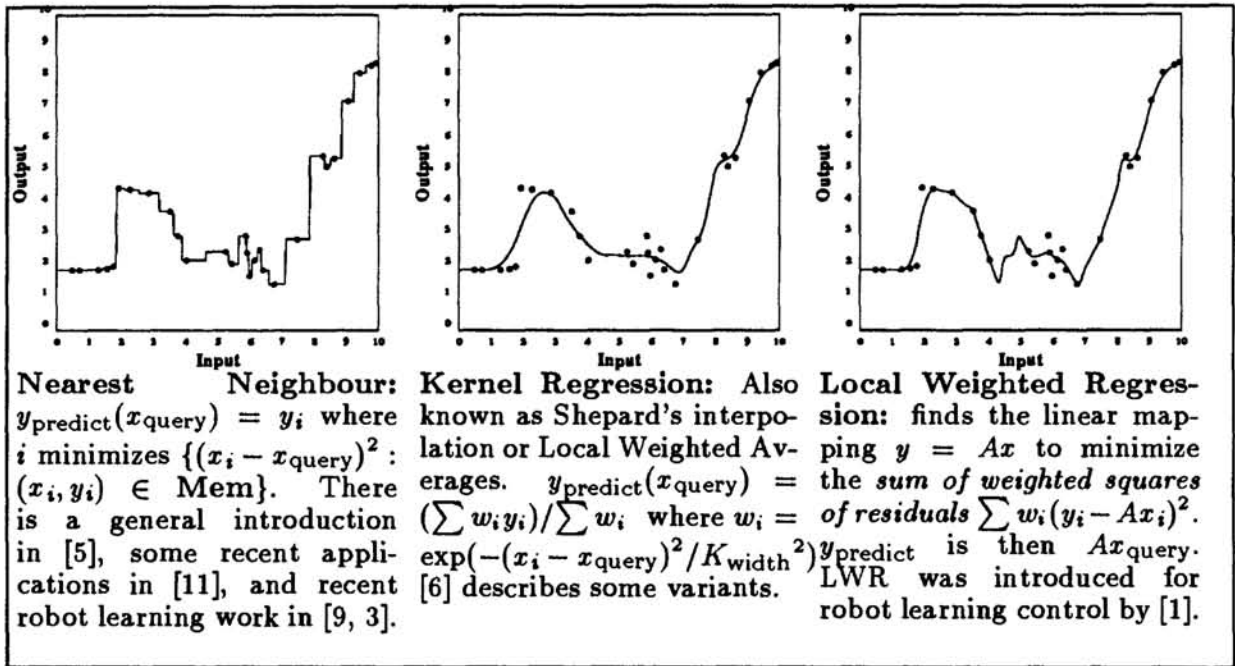

**Nearest Neighbour:** $y_{\text{predict}}(x_{\text{query}}) = y_i$ where $i$ minimizes $\{(x_i - x_{\text{query}})^2 : (x_i, y_i) \in \text{Mem}\}$. There is a general introduction in [5], some recent applications in [11], and recent robot learning work in [9, 3].

**Kernel Regression:** Also known as Shepard's interpolation or Local Weighted Averages. $y_{\text{predict}}(x_{\text{query}}) = (\sum w_i y_i) / \sum w_i$ where $w_i = \exp(-(x_i - x_{\text{query}})^2 / K_{\text{width}}^2)$ [6] describes some variants.

**Local Weighted Regression:** finds the linear mapping $y = Ax$ to minimize the *sum of weighted squares of residuals* $\sum w_i (y_i - Ax_i)^2$. $y_{\text{predict}}$ is then $Ax_{\text{query}}$. LWR was introduced for robot learning control by [1].

## 2   A MEMORY-BASED INVERSE MODEL

An *inverse model* maps **State** × **Behaviour** → **Action** (s × b → a). Behaviour is the output of the system, typically the next state or time derivative of state. The learned inverse model provides a conceptually simple controller:

1. Observe s and $b_{\text{goal}}$.
2. a := inverse-model(s, $b_{\text{goal}}$)
3. Perform action a and observe actual behaviour $b_{\text{actual}}$.
4. Update MEM with (s, $b_{\text{actual}}$ → a): If we are ever again in state s and require behaviour $b_{\text{actual}}$ we should apply action a.

Memory-based versions of this simple algorithm have used nearest neighbour [9] and LWR [3]. $b_{\text{goal}}$ is the goal behaviour: depending on the task it may be fixed or it may vary between control cycles, perhaps as a function of state or time. The algorithm provides aggressive learning: during repeated attempts to achieve the same goal behaviour, the action which is applied is not an incrementally adjusted version of the previous action, but is instead the action which the memory and the memory-based learner predicts will directly achieve the required behaviour. If the function is locally linear then the sequence of actions which are chosen are closely related to the Secant method [4] for numerically finding the zero of a function by bisecting the line between the closest approximations that bracket the $y = 0$ axis. If learning begins with an initial error $E_0$ in the action choice, and we wish to reduce this error to $E_0/K$, the number of learning steps is $O(\log \log K)$: subject to benign conditions, the learner jumps to actions close to the ideal action very quickly.

A common objection to learning the inverse model is that it may be ill-defined. For a memory-based method the problems are particularly serious because of its update rule. It updates the inverse model near $b_{\text{actual}}$ and therefore in those cases in which $b_{\text{goal}}$ and $b_{\text{actual}}$ differ greatly, the mapping near $b_{\text{goal}}$ may not change. As a result,

subsequent cycles will make identical mistakes. [10] discusses this further.

# 3    A MEMORY-BASED FORWARD MODEL

One fix for the problem of inverses becoming stuck is the addition of random noise to actions prior to their application. However, this can result in a large proportion of control cycles being wasted on experiments which the robot should have been able to predict as valueless, defeating the initial aim of learning as quickly as possible.

An alternative technique using multilayer neural nets has been to learn a forward model, which is necessarily well defined, to train a partial inverse. Updates to the forward model are obtained by standard supervised training, but updates to the inverse model are more sophisticated. The local Jacobian of the forward model is obtained and this value is used to drive an incremental change to the inverse model [8]. In conjunction with memory-based methods such an approach has the disadvantage that incremental changes to the inverse model loses the one-shot learning behaviour, and introduces the danger of becoming trapped in a local minimum.

Instead, this investigation only relies on learning the forward model. Then the inverse model is implicitly obtained from it by online numerical inversion instead of direct lookup. This is illustrated by the following algorithm:

1. Observe $s$ and $b_{goal}$.
2. **Perform numerical inversion:**

   Search among a series of candidate actions
   $a_1, a_2 \ldots a_k$:
   $b_1^{predict} := \texttt{forward-model}(s, a_1, MEM)$
   $b_2^{predict} := \texttt{forward-model}(s, a_2, MEM)$
   $\vdots$
   $b_k^{predict} := \texttt{forward-model}(s, a_k, MEM)$

   Until $\boxed{\text{TIME-OUT}}$
   or $\boxed{b_k^{predict} = b_{goal}}$

3. If TIME-OUT then perform experimental action else perform $a_k$.
4. Update MEM with $(s, a_k \rightarrow b_{actual})$

A nice feature of this method is the absence of a preliminary training phase such as random flailing or feedback control. A variety of search techniques for numerical inversion can be applied. Global random search avoids local minima but is very slow for obtaining accurate actions, hill climbing is a robust local procedure and more aggressive procedures such as Newton's method can use partial derivative estimates from the forward model to make large second-order steps. The implementation used for subsequent results had a combination of global search and local hill climbing.

In very high speed applications in which there is only time to make a small number of forward model predictions, it is not difficult to regain much of the speed advantage of directly using an inverse model by commencing the action search with $a_0$ as the action predicted by a learned inverse model.

# 4    OTHER CONSIDERATIONS

Actions selected by a forward memory-based learner can be expected to converge very quickly to the correct action in benign cases, and will not become stuck in difficult cases, provided that the memory based representation can fit the true forward

model. This proviso is weak compared with incremental learning control techniques which typically require stronger prior assumptions about the environment, such as near-linearity, or that an iterative function approximation procedure will avoid local minima. One-shot methods have an advantage in terms of number of control cycles before adequate performance whereas incremental methods have the advantage of only requiring trivial amounts of computation per cycle. However, the simple memory-based formalism described so far suffers from two major problems which some forms of adaptive and neural controllers may avoid.

- Brittle behaviour in the presence of outliers.
- Poor resistance to non-stationary environments.

Many incremental methods implicitly forget all experiences beyond a certain horizon. For example, in the delta rule $\Delta w_{ij} = \nu(y_i^{\text{actual}} - y_i^{\text{predict}})x_j$, the age beyond which experiences have a negligible effect is determined by the learning rate $\nu$. As a result, the detrimental effect of misleading experiences is present for only a fixed amount of time and then fades away[1]. In contrast, memory-based methods remember everything for ever. Fortunately, two statistical techniques: *robust regression* and *cross-validation* allow extensions to the numerical inversion method in which we can have our cake and eat it too.

## 5    USING ROBUST REGRESSION

We can judge the quality of each experience $(x_i, y_i) \in \mathbf{Mem}$ by how well it is predicted by the rest of the experiences. A simple measure of the $i$th error is the *cross validation* error, in which the experience is first removed from the memory before prediction. $e_i^{\text{xve}} = | \mathbf{Predict}(x_i, \mathbf{Mem} - \{(x_i, y_i)\}) |$. With the memory-based formalism, in which all work takes place at prediction time, it is no more expensive to predict a value with one datapoint removed than with it included.

Once we have the measure $e_i^{\text{xve}}$ of the quality of each experience, we can decide if it is worth keeping. Robust statistics [7] offers a wide range of methods: this implementation uses the *Median Absolute Deviation* (MAD) procedure.

## 6    FULL CROSS VALIDATION

The value $e_{\text{total}}^{\text{xve}} = \sum e_i^{\text{xve}}$, summed over all "good" experiences, provides a measure of how well the current representation fits the data. By optimizing this value with respect to internal learner parameters, such as the width of the local weighting function $K_{\text{width}}$ used by kernel regression and LWR, the internal parameters can be found automatically. Another important set of parameters that can be optimized is the relative scaling of each input variable: an example of this procedure applied to a two-joint arm task may be found in Reference [2]. A useful feature of this procedure is its quick discovery (and subsequent ignoring) of irrelevant input variables.

Cross-validation can also be used to selectively forget old inaccurate experiences caused by a slowly drifting or suddenly changing environment. We have already seen that adaptive control algorithms such as the LMS rule can avoid such problems because the effects of experiences decay with time. Memory based methods can also forget things according to a forgetfulness parameter: all observations are weighted

by not only the distance to the $x_{\text{query}}$ but also by their age:

$$w_i = exp(-(x_i - x_{\text{query}})^2/K_{\text{width}}^2 - (n-i)/K_{\text{recall}}) \qquad (1)$$

where we assume the ordering of the experiences' indices $i$ is temporal, with experience $n$ the most recent.

We find the $K_{\text{recall}}$ that minimizes the recent weighted average cross validation error $\sum_{i=0}^{n} e_i^{\text{xve}} \exp(-(n-i)/\gamma)$, where $\gamma$ is a human assigned 'meta-forgetfulness' constant, reflecting how many experiences the learner would need in order to benefit from observation of an environmental change. It should be noted that $\gamma$ is a substantially less task dependent prescription of how far back to forget than would be a human specified $K_{\text{recall}}$. Some initial tests of this technique are included among the experiments of Section 8.

Architecture selection is another use of cross validation. Given a family of learners, the member with the least cross validation error is used for subsequent predictions.

## 7   COMPUTATIONAL CONSIDERATIONS

Unless the real time between control cycles is longer than a few seconds, cross validation is too expensive to perform after every cycle. Instead it can be performed as a separate parallel process, updating the best parameter values and removing outliers every few real control cycles. The usefulness of breaking a learning control task into an online realtime processes and offline mental simulation was noted by [12]. Initially, the small number of experiences means that cross validation optimizes the parameters very frequently, but the time between updates increases with the memory size. The decreasing frequency of cross validation updates is little cause for concern, because as time progresses, the estimated optimal parameter values are expected to become decreasingly variable.

If there is no time to make more than one memory based query per cycle, then memory based learning can nevertheless proceed by pushing even more of the computation into the offline component. If the offline process can identify meaningful states relevant to the task, then it can compute, for each of them, what the optimal action would be. The resulting state-action pairs are then used as a policy. The online process then need only look up the recommended action in the policy, apply it and then insert $(s, a, b)$ into the memory.

## 8   COMPARATIVE TESTS

The ultimate goal of the investigation is to produce a learning control algorithm which can learn to control a fairly wide family of different tasks. Some basic, very different, tasks have been used for the initial tests.

The HARD task, graphed in Figure 1, is a one-dimensional direct relationship between action and behaviour which is both non-monotonic and discontinuous. The VARIER task (Figure 2) is a sinusoidal relation for which the phase continuously drifts, and occasionally alters catastrophically.

LINEAR is a noisy linear relation between 4-d states, 4-d actions and 4-d behaviours. For these first three tasks, the goal behaviour is selected randomly on each control cycle. ARM (Figure 3) is a simulated noisy dynamic two-joint arm acting under gravity in which state is perceived in cartesian coordinates and actions are produced

in joint-torque coordinates. Its task is to follow the circular trajectory. BILLIARDS is a simulation of the real billiards robot described shortly in which 5% of experiences are entirely random outliers.

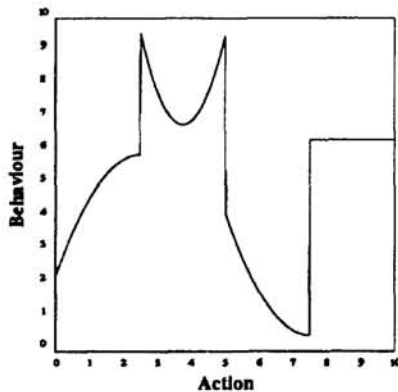
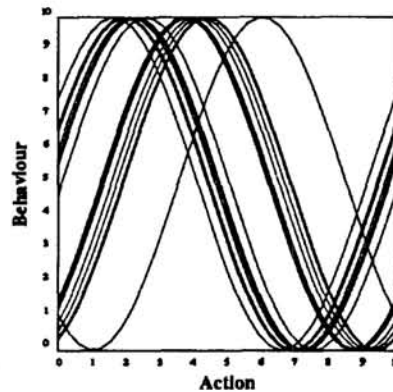
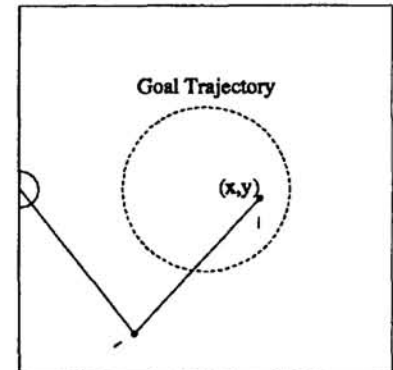

Figure 1: The HARD relation.    Figure 2: VARIER relation.    Figure 3: The ARM task.

The following learning methods were tested: nearest neighbour, kernel regression and LWR, all searching the forward model and using a form of uncertainty-based intelligent experimentation [10] when the forward search proved inadequate. Another method under test was sole use of the inverse, learned by LWR. Finally a "best-possible" value was obtained by numerically inverting the real simulated forward model instead of a learned model.

All tasks were run for only 200 control cycles. In each case the quality of the learner was measured by the number of successful actions in the final hundred cycles, where "successful" was defined as producing behaviour within a small tolerance of $b_{goal}$.

Results are displayed in Table 1. There is little space to discuss them in detail, but they generally support the arguments of the previous sections. The inverse model on its own was generally inferior to the forward method, even in those cases in which the inverse is well-defined. Outlier removal improved performance on the BILLIARDS task over non-robustified versions. Interestingly, outlier removal also greatly benefited the inverse only method. The selectively forgetful methods performed better than than their non-forgetful counterparts on the VARIER task, but in the stationary environments they did not pay a great penalty. Cross validation for $K_{width}$ was useful: for the HARD task, LWR found a very small $K_{width}$ but in the LINEAR task it unsurprisingly preferred an enormous $K_{width}$.

Some experiments were also performed with a real billiards robot shown in Figure 4. Sensing is visual: one camera looks along the cue stick and the other looks down at the table. The cue stick swivels around the cue ball, which starts each shot at the same position. At the start of each attempt the object ball is placed at a random position in the half of the table opposite the cue stick. The camera above the table obtains the $(x, y)$ image coordinates of the object ball, which constitute the state. The action is the $x$-coordinate of the image of the object ball on the cue stick camera. A motor swivels the cue stick until the centroid of the actual image of the object ball coincides with the chosen $x$-coordinate value. The shot is then performed and observed by the overhead camera. The behaviour is defined as the cushion and position on the cushion with which the object ball first collides.

| Controller type. (K = use MAD outlier removal, X = use cross-validation for $K_{\text{width}}$, R = use cross-validation for $K_{\text{recall}}$, IF = obtain initial candidate action from the inverse model then search the forward model.) | VARIER | HARD | LINEAR | ARM | BIL'DS |
|---|---|---|---|---|---|
| Best Possible: Obtained from numerically inverting simulated world | $100 \pm 0$ | $100 \pm 0$ | $75 \pm 3$ | $94 \pm 1$ | $82 \pm 4$ |
| Inverse only, learned with LWR | $15 \pm 9$ | $24 \pm 11$ | $7 \pm 6$ | $76 \pm 28$ | $71 \pm 5$ |
| Inverse only, learned with LWR, KRX | $48 \pm 16$ | $72 \pm 8$ | $70 \pm 4$ | $89 \pm 4$ | $70 \pm 10$ |
| LWR: IF | $14 \pm 10$ | $11 \pm 5$ | $58 \pm 4$ | $83 \pm 4$ | $55 \pm 12$ |
| LWR: IF X | $19 \pm 9$ | $72 \pm 4$ | $70 \pm 4$ | $89 \pm 3$ | $61 \pm 9$ |
| LWR: IF KX | $22 \pm 15$ | $51 \pm 27$ | $73 \pm 3$ | $90 \pm 3$ | $75 \pm 7$ |
| LWR: IF KRX | $54 \pm 8$ | $65 \pm 28$ | $70 \pm 5$ | $89 \pm 2$ | $69 \pm 7$ |
| LWR: Forward only, KRX | $56 \pm 9$ | $53 \pm 17$ | $73 \pm 1$ | $89 \pm 1$ | $69 \pm 7$ |
| Kernel Regression: IF | $8 \pm 2$ | $6 \pm 2$ | $13 \pm 3$ | $3 \pm 2$ | $1 \pm 1$ |
| Kernel Regression: IF KRX | $15 \pm 8$ | $42 \pm 21$ | $14 \pm 2$ | $23 \pm 10$ | $30 \pm 5$ |
| Nearest Neighbour: IF | $22 \pm 4$ | $92 \pm 2$ | $0 \pm 0$ | $44 \pm 6$ | $10 \pm 2$ |
| Nearest Neighbour: IF K | $26 \pm 10$ | $69 \pm 4$ | $0 \pm 0$ | $40 \pm 6$ | $9 \pm 3$ |
| Nearest Neighbour: IF KR | $44 \pm 8$ | $68 \pm 3$ | $0 \pm 0$ | $40 \pm 7$ | $11 \pm 3$ |
| Nearest Neighbour: Forward only, KR | $43 \pm 8$ | $66 \pm 5$ | $0 \pm 0$ | $37 \pm 3$ | $8 \pm 1$ |
| Global Linear Regression: IF | $8 \pm 3$ | $7 \pm 3$ | $74 \pm 5$ | $60 \pm 17$ | $23 \pm 6$ |
| Global Linear Regression: IF KR | $20 \pm 13$ | $9 \pm 2$ | $73 \pm 4$ | $72 \pm 3$ | $21 \pm 4$ |
| Global Quadratic Regression: IF | $14 \pm 7$ | $5 \pm 3$ | $64 \pm 2$ | $70 \pm 22$ | $40 \pm 11$ |

Table 1: Relative performance of a family of learners on a family of tasks. Each combination of learner and task was run ten times to provide the mean number of successes and standard deviation shown in the table.

The controller uses the memory based learner to choose the action to maximize the probability that the ball will enter the nearer of the two pockets at the end of the table. A histogram of the number of successes against trial number is shown in Figure 5. In this experiment, the learner was LWR using outlier removal and cross validation for $K_{\text{width}}$. After 100 experiences, control choice running on a Sun-4 was taking 0.8 seconds[2]. Sinking the ball requires better than 1% accuracy in the choice of action, the world contains discontinuities and there are random outliers in the data and so it is encouraging that within less than 100 experiences the robot had reached a 70% success rate—substantially better than the author can achieve.

## ACKNOWLEDGEMENTS

Some of the work discussed in this paper is being performed in collaboration with Chris Atkeson. The robot cue stick was designed and built by Wes Huang with help from Gerrit van Zyl. Dan Hill also helped considerably with the billiards robot. The author is supported by a Postdoctoral Fellowship from SERC/NATO. Support was provided under Air Force Office of Scientific Research grant AFOSR-89-0500 and a National Science Foundation Presidential Young Investigator Award to Christopher G. Atkeson.

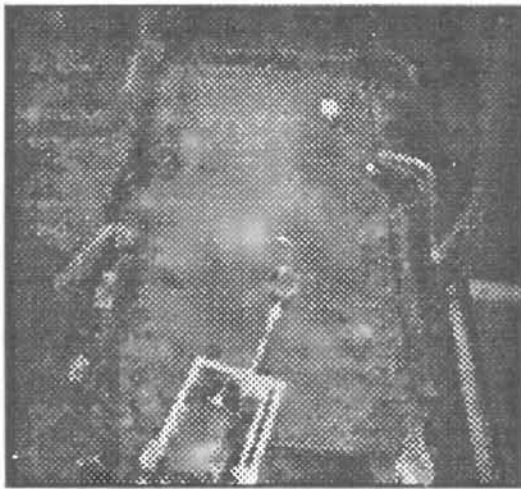

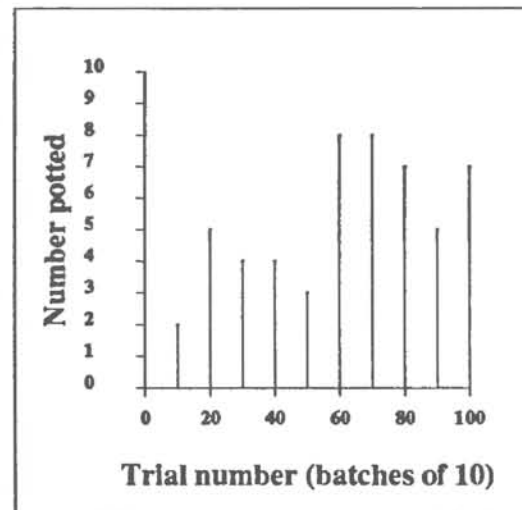

Figure 4: The billiards robot. In the foreground is the cue stick which attempts to sink balls in the far pockets.

Figure 5: Frequency of successes versus control cycle for the billiards task.

## Footnotes

[1]This also has disadvantages: persistence of excitation is required and multiple tasks can often require relearning if they have not been practised recently.

[2]This could have been greatly improved with more appropriate hardware or better software techniques such as $k$d-trees for structuring data [11, 9].

# References

[1] C. G. Atkeson. Using Local Models to Control Movement. In *Proceedings of Neural Information Processing Systems Conference*, November 1989.

[2] C. G. Atkeson. Memory-Based Approaches to Approximating Continuous Functions. Technical report, M. I. T. Artificial Intelligence Laboratory, 1990.

[3] C. G. Atkeson and D. J. Reinkensmeyer. Using Associative Content-Addressable Memories to Control Robots. In Miller, Sutton, and Werbos, editors, *Neural Networks for Control*. MIT Press, 1989.

[4] S. D. Conte and C. De Boor. *Elementary Numerical Analysis*. McGraw Hill, 1980.

[5] R. O. Duda and P. E. Hart. *Pattern Classification and Scene Analysis*. John Wiley & Sons, 1973.

[6] R. Franke. Scattered Data Interpolation: Tests of Some Methods. *Mathematics of Computation*, 38(157), January 1982.

[7] F. Hampbell, P. Rousseeuw, E. Ronchetti, and W. Stahel. *Robust Statistics*. Wiley International, 1985.

[8] M. I. Jordan and D. E. Rumelhart. Forward Models: Supervised Learning with a Distal Teacher. Technical report, M. I. T., July 1990.

[9] A. W. Moore. Efficient Memory-based Learning for Robot Control. PhD. Thesis; Technical Report No. 209, Computer Laboratory, University of Cambridge, October 1990.

[10] A. W. Moore. Knowledge of Knowledge and Intelligent Experimentation for Learning Control. In *Proceedings of the 1991 Seattle International Joint Conference on Neural Networks*, July 1991.

[11] S. M. Omohundro. Efficient Algorithms with Neural Network Behaviour. *Journal of Complex Systems*, 1(2):273–347, 1987.

[12] R. S. Sutton. Integrated Architecture for Learning, Planning, and Reacting Based on Approximating Dynamic Programming. In *Proceedings of the 7th International Conference on Machine Learning*. Morgan Kaufman, June 1990.
